# Analogy--Watershed or Waterloo?
# Structural alignment and the development of
# connectionist models of analogy

**Dedre Gentner**
Department of Psychology
Northwestern University
2029 Sheridan Rd.
Evanston, IL 60208

**Arthur B. Markman**
Department of Psychology
Northwestern University
2029 Sheridan Rd.
Evanston, IL 60208

## ABSTRACT

Neural network models have been criticized for their inability to make use of compositional representations. In this paper, we describe a series of psychological phenomena that demonstrate the role of structured representations in cognition. These findings suggest that people compare relational representations via a process of structural alignment. This process will have to be captured by any model of cognition, symbolic or subsymbolic.

## 1.0 INTRODUCTION

Pattern recognition is central to cognition. At the perceptual level, we notice key features of the world (like symmetry), recognize objects in front of us and identify the letters on a printed page. At a higher level, we recognize problems we have solved before and determine similarities—including analogical similarities—between new situations and old ones. Neural network models have been successful at capturing sensory pattern recognition (e.g., Sabourin & Mitiche, 1992). In contrast, the determination of higher level similarities has been well modeled by symbolic processes (Falkenhainer, Forbus, & Gentner, 1989). An important question is whether neural networks can be extended to high-level similarity and pattern recognition.

In this paper, we will summarize the constraints on cognitive representations suggested by the psychological study of similarity and analogy. We focus on three themes: (1) structural alignment; (2) structural projection; and (3) flexibility.

## 2.0  STRUCTURAL ALIGNMENT IN SIMILARITY

Extensive psychological research has examined the way people compare pairs of items to determine their similarity.  Mounting evidence suggests that the similarity of two complex items depends on the degree of match between their component objects (common and distinctive *attributes*) and on the degree of match between the *relations* among the component objects.  Specifically, there is evidence that (1) similarity involves structured pattern matching, (2) similarity involves structured pattern completion, (3) comparing the same item with different things can highlight different aspects of the item and (4) even comparisons of a single pair of items may yield multiple interpretations.  We will examine these four claims in the following sections.

## 2.1  SIMILARITY INVOLVES STRUCTURED PATTERN MATCHING

The central idea underlying structured pattern matching is that similarity involves an alignment of relational structure.  For example, in Figure 1a, configuration A is clearly more similar to the top configuration than configuration B, because A has similar objects taking part in the same relation (*above*), while B has similar objects taking part in a different relation (*next-to*).  This determination can be made regardless of whether the objects taking part in the relations are similar.  For example, in Figure 1b configuration A is also more similar to the top configuration than is configuration B, because A shares a relation with the top configuration, while B does not.  As a check on this intuition, 10 subjects were asked to tell us which configuration (A or B) went best with the top configuration for the triads in Figures 1a and 1b.  All 10 subjects chose configuration A for both triads.  This example demonstrates that relations (such as the common *above* relation) are important in similarity processing.

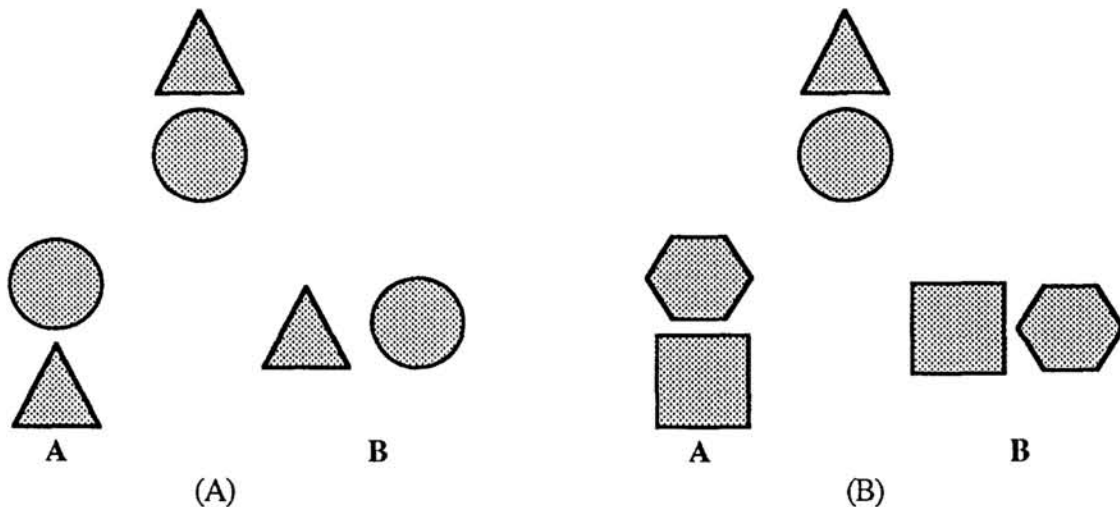

**Figure 1.** Examples of structural alignment in perception.

The importance of relations was also demonstrated by Palmer (1978) who asked subjects to rate the similarity of pairs of configurations like those in Figure 2.  The pair in Figure 2a shares the global property that both are open figures, while the pair in Figure 2b does not.  As would be expected if subjects attend to relations when determining similarity,

higher similarity ratings were given to pairs like the one in Figure 2a than to pairs like the one in Figure 2b. This finding can only be explained by appealing to structural similarity, because both pairs of configurations share the same number of local line segments. Consistent with this result, Palmer also found that subjects were faster to say that the items in Figure 2b are different than that the items in Figure 2a are different. A similar result was obtained by Lockhead and King (1977).

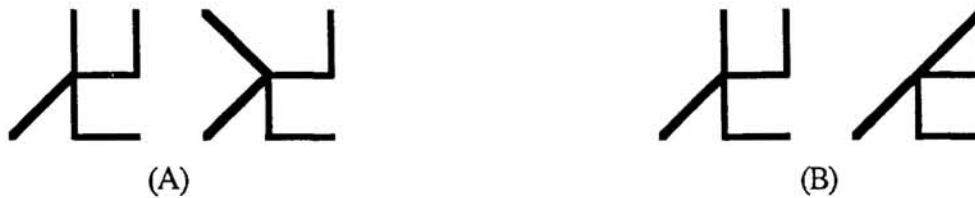

(A)                                                    (B)

**Figure 2**. Structured pattern matching in a study by Palmer (1978).

Further research suggests that common bindings between relations and the items they relate are also central to similarity. For example, Clement and Gentner (1991) presented subjects with pairs of analogous stories. One story described organisms called Tams that ate rocks, while the other described robots that collected data on a planet. In each story, one matching fact also had a matching causal antecedent. For example, *the Tams' exhausting the minerals on the rock CAUSED them to move to another rock*, while *the robots' exhausting the data on a planet CAUSED them to move to another planet*. A second matching fact did not have a matching causal antecedent. For example, *the Tams' underbelly could not function on a new rock* and the *robots' probe could not function on a new planet*, but the causes of these facts did not match. Subjects were asked which of the two pairs of key facts (shown in bold) was more important to the stories. Subjects selected the pair that had the matching causal antecedent, suggesting that their mappings preserved the relational connections in the stories.

## 2.2   STRUCTURED PATTERN COMPLETION

Pattern completion has long been a central feature of neural network models (Anderson, Silverstein, Ritz, & Jones, 1977; Hopfield, 1982). For example, in the BSB model of Anderson et al., vectors in which some units are below their maximum value are filled in by completing a pattern based on the vector similarities of the current activation pattern to previously learned patterns.

The key issue here is the kind of information that guides pattern completion in humans. Data from psychological studies suggests that subjects' pattern matching ability is controlled by structural similarities rather than by geometric similarities. For example, Medin and Goldstone presented subjects with pairs of objects like those in Figure 3 (Medin, Goldstone & Gentner, in press). The left-hand figure in both pairs is somewhat ambiguous, but the right-hand figure is not. Subjects who were asked to list the commonalities of the pair in Figure 3a said that both figures had three prongs, while subjects who were asked to list the commonalities of the pair in Figure 3b said that both figures had four prongs. This finding was obtained for 15 of 16 triads tested, and suggests

that subjects were mapping the structure from the unambiguous figure onto the ambiguous one. Of course, in order for the mapping to take place, the underlying structure of the figures has to be readily alignable, and there must be ambiguity in the target figure. In the pair in Figure 3c, the left hand item cannot be viewed as having four prongs, and so this mapping is not made.

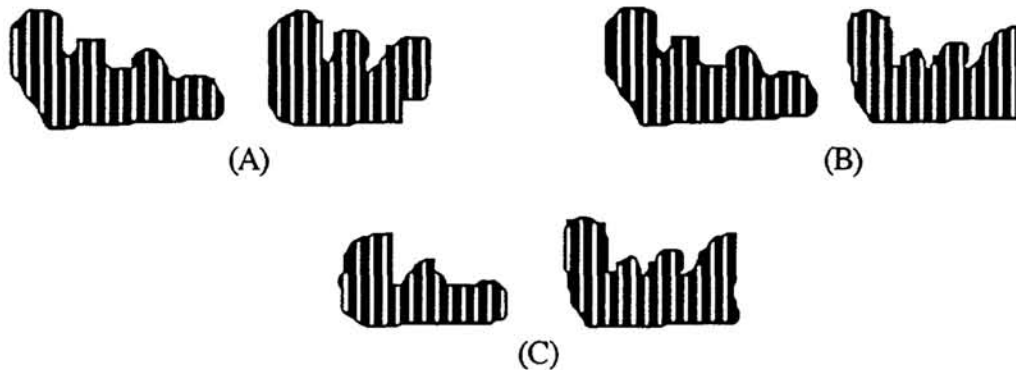

(A)                                                    (B)

(C)

**Figure 3:** Example of structured pattern completion.

Structured pattern completion also occurs in conceptual structures. Clement and Gentner (1991) extended the study described above by deleting the key matching facts from one of the stories (e.g., the **bold** facts from the robot story). Subjects read both stories, then predicted one new fact about the robot story. Subjects were free to predict anything at all, but 50% of the subjects predicted the fact with the matching causal antecedent, while only 28% of the subjects predicted the fact with no matching causal antecedent. By comparison, a control group that made predictions about the target story without seeing the base predicted both facts at the same rate (about 5%). This finding underlines the importance of connectivity in pattern completion. People's predictions were determined not just by the local information, but by whether it was connected to matching information. Thus, pattern completion is structure-sensitive.

## 2.3 DIFFERENT COMPARISONS-DIFFERENT INTERPRETATIONS

Comparison is flexible. When an item takes part in many comparisons, it may be interpreted differently in each comparison. For example, in Figure 3a, the left figure is interpreted as having 3 prongs, while in Figure 3b, it is interpreted as having 4 prongs. Similarly, the comparison 'My surgeon is a butcher' conveys a clumsy surgeon, but 'Genghis Khan was a butcher' conveys a ruthless killer (Glucksberg and Keysar, 1992).

This type of flexibility is also evident in an example presented by Spellman and Holyoak (1992). They pointed out that some politicians likened the Gulf War to World War II, implying that the United States was acting as the world's policeman to stop a tyrant. Other politicians compared Operation Desert Storm to Vietnam, implying that the United States entered into a potentially endless conflict between two other nations. Clearly, different comparisons highlighted different features of the Gulf War.

## 2.4  SAME COMPARISON-DIFFERENT INTERPRETATIONS

Even a single comparison can yield more than one distinct interpretation. This situation may arise when the items are richly represented, with many different clusters of knowledge. It can also arise when the comparison permits more than one alignment, as when the similarities of the objects in an item suggest different correspondences than do the relational similarities (i.e. components are *cross-mapped* (Gentner & Toupin, 1986)).

Markman and Gentner (in press) presented subjects with pairs of scenes like those depicting the perceptual higher order relation *monotonic increase in size* shown in Figure 4. In Figure 4, the circle with the arrow over it in the left-hand figure is the largest circle in the array. It is cross-mapped, since it is the same size as the middle circle in the right-hand figure, but plays the same relational role as the left (largest) circle. Subjects were given a mapping task in which they were asked to point to the object in the right-hand figure that went with with the cross-mapped circle in the left-hand figure. In this task, subjects chose the circle that looked most similar 91% of the time. However, a second group of subjects, who rated the similarity of the pair before doing this mapping, selected the object playing the same relational role 61% of the time. In both tasks, when subjects were asked whether there were any other good choices, they generally described the other possible mapping. These results show that the same comparison can be aligned in different ways, and that similarity comparisons promote structural alignment.

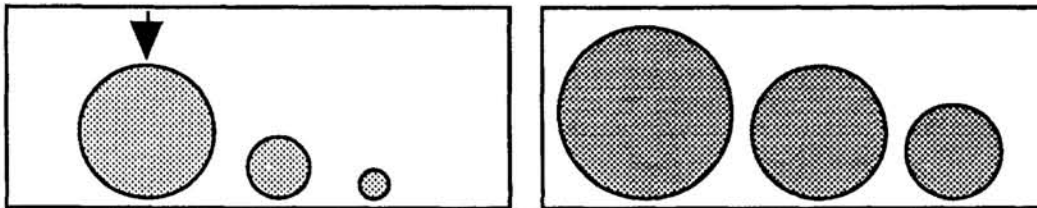

**Figure 4:** Stimuli with a cross-mapping from Markman and Gentner (in press).

Goldstone (personal communication) has demonstrated that, not only are comparisons flexible, but subjects can attend to attribute and relation matches selectively. He presented subjects with triads like the one in Figure 5. Subjects were asked to choose either the bottom figure with the most attribute similarity to the top one, or the bottom figure with the most relational similarity to the top one. In this study, and other pilot studies, subjects were highly accurate at both task, suggesting flexibility to attend to different kinds of similarity.

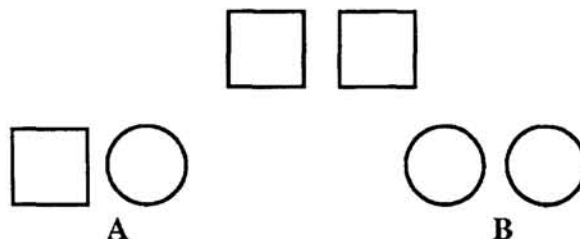

**Figure 5:** Sample stimuli from study by Goldstone.

Similar flexibility can also be found in stimuli with conceptual relations. Gentner (1988) presented children with *double metaphors* that can have two meanings, one based on attribute similarities and a second based on relational similarities. For example, the metaphor 'Plant stems are like drinking straws' can mean that both are round and skinny, or that both transport fluids from low places to high places. Gentner found that young children (age 5-6) made the attribute-based interpretation, while older children (age 9-10) and adults could make either interpretation (but preferred the relation-based interpretation).

There are limits to this flexibility. People prefer to make structurally consistent mappings (Gentner, 1983). For example, Spellman and Holyoak (1992) told subjects to map Operation Desert Storm onto World War II. When they asked subjects to find a correspondence for George Bush given that Saddam Hussein corresponded to Hitler, subjects generally chose either FDR or Churchill. Then, subjects were asked to make a mapping for the United States in 1991. Interestingly, subjects who mapped Bush to FDR almost always mapped the US in 1991 to the US during World War II. In contrast, subjects who mapped Bush to Churchill almost always mapped the US in 1991 to Britain during World War II. Thus, subjects maintained structurally consistent mappings.

This type of flexibility adds significant complexity to the comparison process, because a system cannot simply be trained to search for relational correspondences or be taught to prefer only attribute matches. Rather, the comparison process must determine both attribute and relation matches and must be able to keep different mappings distinct from each other.

## 2.5 SUMMARY OF EMPIRICAL EVIDENCE

These findings suggest that comparisons of both perceptual and conceptual materials involve structural alignment. Further, structural alignment promotes structure sensitive pattern completion. Finally, comparisons allow for multiple interpretations of a single item in different comparisons or multiple interpretations of a single comparison. Any model of human cognition that involves comparison must exhibit these properties.

## 3.0 IMPLICATIONS FOR COGNITIVE MODELS

Many of the questions concerning the adequacy of connectionist models and neural networks for high-level cognitive tasks have centered on linguistic processing and the crucial role of compositional relational structures in sentence comprehension (Fodor & McLaughlin, 1990; Fodor & Pylyshyn, 1988). Recent work has addressed this problem by examining ways to represent hierarchical structure in connectionist models, implementing stacks and binary trees to model variable binding and recursive sentence processing (e.g., Elman, 1990; Pollack, 1990; Smolensky, 1990; see also Quinlan, 1991 for a review). It is too soon to tell how successful these methods will be, or whether they can be extended to the general case of structural alignment.

The results summarized here underline the need for representations that permit structural alignment. How should this be done? As van Gelder (1990) discusses, symbolic systems traditionally use *concatenative* representation, in which symbol names are concatenated to build a compositional representation. For example, a circle above a triangle could be

represented by the assertion **above**(circle,triangle). Such symbolic representations have been used to model the analogy and similarity phenomena described here with some success (Falkenhainer, et al., 1989). Van Gelder (1990) suggests a weaker criterion of *functional compositionality*. In functionally compositional representations, tokens for the symbols are not directly present in the representation, but they can be extracted from the representation via some process. Van Gelder suggests that the natural representation used by neural networks is functionally compositional. Analogously, the question of whether connectionist models can model the phenomena described here should be couched in terms of *functional alignability*: whether the representations can be decomposed and aligned, rather than whether the structure is transparently present.

Along this track, an intriguing question is whether the surface form of functionally compositional representations will be similar to the degree that the structures they represent are similar. If so, the alignment process could take place simply by comparing activation vectors. As yet, there are no networks that exhibit this behavior. Further, given the evidence that geometric representations are insufficient to model human similarity comparisons (see Tversky (1977) for a review), we are pessimistic about the prospects that this type of model will be developed.

In conclusion, substantial psychological evidence suggests that determining the similarity of two items requires a flexible alignment of structured representations. We suspect that connectionist models of cognitive processes that involve comparisons will have to exhibit concatenative compositionality in order to capture the flexibility inherent in comparisons. However, we leave open the possibility that systems exhibiting functional alignability will be successful.

## Acknowledgments

This research was sponsored by ONR grant BNS-87-20301. We thank Jon Handler, Ed Wisniewski, Phil Wolff and the whole Similarity and Analogy group for comments on this work. We also thank Laura Kotovsky, Catherine Kreiser and Russ Poldrack for running the pilot studies described above.

## References

Anderson, J. A., Silverstein, J. W., Ritz, S. A., & Jones, R. S. (1977). Distinctive features, categorical perception and probability learning: Some applications of a neural model. Psychological Review, 84, 413-451.

Clement, C. A., & Gentner, D. (1991). Systematicity as a selection constraint in analogical mapping. Cognitive Science, 15, 89-132.

Elman, J.L. (1990). Finding structure in time. Cognitive Science, 14(2), 179-212.

Falkenhainer, B., Forbus, K. D., & Gentner, D. (1989). The structure-mapping engine: Algorithm and examples. Artificial Intelligence, 41(1), 1-63.

Fodor, J., & McLaughlin, B. (1990). Connectionism and the problem of systematicity: Why Smolensky's solution doesn't work. Cognition, 35, 183-204.

Fodor, J. A., & Pylyshyn, Z. W. (1988). Connectionism and cognitive architecture:  A critical analysis. Cognition, 28, 3-71.

Gentner, D. (1983). Structure mapping:  A theoretical framework for analogy. Cognitive Science, 7, 155-170.

Gentner, D. (1988). Metaphor as structure mapping:  The relational shift. Child Development, 59, 47-59.

Gentner, D., & Toupin, C. (1986). Systematicity and surface similarity in the development of analogy. Cognitive Science, 10, 277-300.

Glucksberg, S. & Keysar, B. (1990). Understanding metaphorical comparisons:  Beyond similarity.  Psychological Review, 97(1), 3-18.

Hopfield, J. J. (1982). Neural networks and physical systems with emergent collective computational abilities. Proceedings of the National Academy of Sciences, 79, 2554-2558.

Lockhead, G. R., & King, M. C. (1977). Classifying integral stimuli. Journal of Experimental Psychology:  Human Perception and Performance, 3(3), 436-443.

Markman, A. B., & Gentner, D. (in press). Structural alignment during similarity comparisons. Cognitive Psychology.

Medin, D. L., Goldstone, R. L., & Gentner, D. (in press). Respects for similarity. Psychological Review.

Palmer, S. E. (1978). Structural aspects of visual similarity. Memory and Cognition, 6(2), 91-97.

Pollack, J. B. (1990). Recursive distributed representations. Artificial Intelligence, 46(1-2), 77-106.

Quinlan, P.T. (1991). Connectionism and Psychology:  A psychological perspective on new connectionist research.  Chicago:  The University of Chicago Press.

Sabourin, M. & Mitiche, A. (1992).  Optical character recognition by a neural network. Neural Networks, 5(5), 843-852.

Smolensky, P. (1990). Tensor product variable binding and the representation of symbolic structures in connectionist systems. Artificial Intelligence, 46, 159-216.

Spellman, B. A., & Holyoak, K. J. (1992). If Saddam is Hitler then who is George Bush?  Analogical mapping between systems of social roles. Journal of Personality and Social Psychology, 62(6), 913-933.

Tversky, A. (1977). Features of similarity. Psychological Review, 84(4), 327-352.

van Gelder, T. (1990). Compositionality:  A connectionist variation on a classical theme. Cognitive Science, 14(3), 355-384.
